# A MODEL OF AUDITORY STREAMING

**Susan L. McCabe & Michael J. Denham**
Neurodynamics Research Group
School of Computing
University of Plymouth
Plymouth PL4 8AA, U.K.

## ABSTRACT

An essential feature of intelligent sensory processing is the ability to focus on the part of the signal of interest against a background of distracting signals, and to be able to direct this focus at will. In this paper the problem of auditory scene segmentation is considered and a model of the early stages of the process is proposed. The behaviour of the model is shown to be in agreement with a number of well known psychophysical results. The principal contribution of this model lies in demonstrating how streaming might result from interactions between the tonotopic patterns of activity of input signals and traces of previous activity which feedback and influence the way in which subsequent signals are processed.

## 1  INTRODUCTION

The appropriate segmentation and grouping of incoming sensory signals is important in enabling an organism to interact effectively with its environment (Llinas, 1991). The formation of associations between signals, which are considered to arise from the same external source, allows the organism to recognise significant patterns and relationships within the signals from each source without being confused by accidental coincidences between unrelated signals (Bregman, 1990). The intrinsically temporal nature of sound means that in addition to being able to focus on the signal of interest, perhaps of equal significance, is the ability to predict how that signal is expected to progress; such expectations can then be used to facilitate further processing of the signal. It is important to remember that perception is a creative act (Luria, 1980). The organism creates its interpretation of the world in response to the current stimuli, within the context of its current state of alertness, attention, and previous experience. The creative aspects of perception are exemplified in the auditory system where peripheral processing decomposes acoustic stimuli. Since the frequency spectra of complex sounds generally

overlap, this poses a complicated problem for the auditory system : which parts of the signal belong together, and which of the subgroups should be associated with each other from one moment to the next, given the extra complication of possible discontinuities and occlusion of sound signals? The process of streaming effectively acts to to associate those sounds emitted from the same source and may be seen as an accomplishment, rather than the breakdown of some integration mechanism (Bregman, 1990).

The cognitive model of streaming, proposed by (Bregman, 1990), is based primarily on Gestalt principles such as common fate, proximity, similarity and good continuation. Streaming is seen as a multistage process, in which an initial, preattentive process partitions the sensory input, causing successive sounds to be associated depending on the relationship between pitch proximity and presentation rate. Further refinement of these sound streams is thought to involve the use of attention and memory in the processing of single streams over longer time spans.

Recently a number of computational models which implement these concepts of streaming have been developed. A model of streaming in which pitch trajectories are used as the basis of sequential grouping is proposed by (Cooke, 1992). In related work, (Brown, 1992) uses data-driven grouping schema to form complex sound groups from frequency components with common periodicity and simultaneous onset. Sequential associations are then developed on the basis of pitch trajectory. An alternative approach suggests that the coherence of activity within networks of coupled oscillators, may be interpreted to indicate both simultaneous and sequential groupings (Wang, 1995), (Brown, 1995), and can, therefore, also model the streaming of complex stimuli. Sounds belonging to the same stream, are distinguished by synchronous activity and the relationship between frequency proximity and stream formation is modelled by the degree of coupling between oscillators.

A model, which adheres closely to auditory physiology, has been proposed by (Beauvois, 1991). Processing is restricted to two frequency channels and the streaming of pure tones. The model uses competitive interactions between frequency channels and leaky integrator model neurons in order to replicate a number of aspects of human psychophysical behaviour. The model, described here, used Beauvois' work as a starting point, but has been extended to include multichannel processing of complex signals. It can account for the relationship streaming and frequency difference and time interval (Beauvois, 1991), the temporal development and variability of streaming perceptions (Anstis, 1985), the influence of background organisation on foreground perceptions (Bregman, 1975), as well as a number of other behavioural results which have been omitted due to space limitations.

## 2   THE MODEL

We assume the existence of tonotopic maps, in which frequency is represented as a distributed pattern of activity across the map. Interactions between the excitatory tonotopic patterns of activity reflecting stimulus input, and the inhibitory tonotopic masking patterns, resulting from previous activity, form the basis of the model. In order to simulate behavioural experiments, the relationship between characteristic frequency and position across the arrays is determined by equal spacing within the ERB scale (Glasberg, 1990). The pattern of activation across the tonotopic axis is represented in terms of a Gaussian function with a time course which reflects the onset-type activity found frequently within the auditory system.

Input signals therefore take the form :

$$i(x, t) = c_1(t - t_{Onset})e^{-c_2(t-t_{Onset})}e^{\frac{-1}{2\alpha^2}(f_c(x)-f_s)^2}$$ [1]

where $i(x,t)$ is the probability of input activity at position $x$, time $t$. $c_1$ and $c_2$ are constants, $t_{Onset}$ is the starting time of the signal, $f_c(x)$ is the characteristic frequency at position $x$, $f_s$ is the stimulus frequency, and $\alpha$ determines the spread of the activation.

In models where competitive interactions within a single network are used to model the streaming process, such as (Beauvois, 1991), it is difficult to see how the organisation of background sounds can be used to improve foreground perceptions (Bregman, 1975) since the strengthening of one stream generally serves to weaken others. To overcome this problem, the model of preattentive streaming proposed here, consists of two interacting networks, the foreground and background networks, F and B; illustrated in figure 1. The output from F indicates the activity, if any, in the foreground, or attended stream, and the output from B reflects any other activity. The interaction between the two eventually ensures that those signals appearing in the output from F, i.e. in the foreground stream, do not appear in the output from B, the background; and vice versa. In the model, strengthening of the organisation of the background sounds, results in the 'sharpening' of the foreground stream due to the enhanced inhibition produced by a more coherent background.

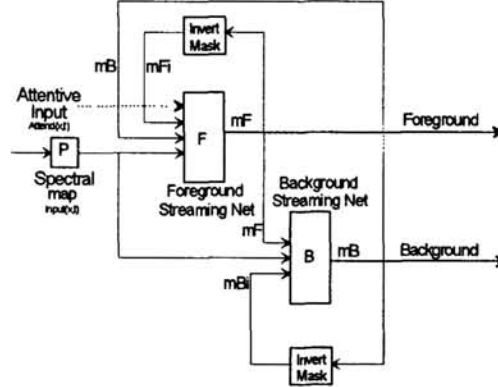

Figure 1 : Connectivity of the Streaming Networks.

Neurons within each array do not interact with each other but simply perform a summation of their input activity. A simplified neuron model with low-pass filtering of the inputs, and output representing the probability of firing, is used :

$$p(x, t) = \sigma[\sum_j v_j(x, t)], \text{ where } \sigma(y) = \frac{1}{1+e^{-y}}$$ [2]

The inputs to the foreground net are :

$$v_1(x, t) = (1 - \frac{dt}{\tau_1})v_1(x, t - dt) + V_1.\phi(i(x, t)).dt$$ [3]

$$v_2(x, t) = (1 - \frac{dt}{\tau_2})v_2(x, t - dt) + V_2.\phi(mFi(x, t - dt)).dt$$ [4]

$$v_3(x, t) = (1 - \frac{dt}{\tau_3})v_3(x, t - dt) + V_3.\phi(mB(x, t - dt)).dt$$ [5]

where $x$ is the position across the array, time $t$, sampling rate $dt$. $\tau_i$ are time constants which determine the rate of decay of activity, $V_i$ are weights on each of the inputs, and $\phi(y)$ is a function used to simulate the stochastic properties of nerve firing which returns a value of $1$ or $0$ with probability $y$.

The output activity pattern in the foreground net and its 'inverse', $mF(x,t)$ and $mFi(x,t)$, are found by :

$$mF(x, t) = \sigma[v_1(x, t) - \eta(v_2(x, t), n) - \eta(v_3(x, t), n)] \qquad [6]$$

$$mFi(x, t) = \max\{[\frac{1}{N} \sum_{i=1}^{N} mF(x_i, t - dt)] - mF(x, t - dt), 0\} \qquad [7]$$

where $\eta(v(x,t),n)$ is the mean of the activity within neighbourhood $n$ of position $x$ at time $t$ and $N$ is the number of frequency channels. Background inputs are similarly calculated.

To summarise, the current activity in response to the acoustic stimulus forms an excitatory input to both the foreground and background streaming arrays, F and B. In addition, F receives inhibitory inputs reflecting the current background activity, and the inverse of the current foreground activity. The interplay between the excitatory and inhibitory activities causes the model to gradually focus the foreground stream and exclude extraneous stimuli. Since the patterns of inhibitory input reflect the distributed patterns of activity in the input, the relationship between frequency difference and streaming, results simply from the graded inhibition produced by these patterns. The relationship between tone presentation rate and streaming is determined by the time constants in the model which can be tuned to alter the rate of decay of activity.

To enable comparisons with psychophysical results, we view the judgement of coherence or streaming made by the model as the difference between the strength of the foreground response to one set of tones compared to the other. The strength of the response to a given frequency, $Resp(f,t)$, is a weighted sum of the activity within a window centred on the frequency :

$$Resp(f, t) = \sum_{i=-W}^{W} mF(x(f) + i, t) * e^{-\frac{i^2}{2\alpha^2}} \qquad [8]$$

where $W$ determines the size of the window centred on position, $x(f)$, the position in the map corresponding to frequency $f$, and $\alpha$ determines the spread of the weighting function about position $x(f)$.

The degree of coherence between two tones, say $f_1$ and $f_2$, is assumed to depend on the difference in strength of foreground response to the two :

$$Coh(f_1, f_2, t) = 1 - \left| \frac{Resp(f_1, t) - Resp(f_2, t)}{Resp(f_1, t) + Resp(f_2, t)} \right| \qquad [9]$$

where $Coh(f_1, f_2, t)$ ranges between 0, when $Resp(f_1, t)$ or $Resp(f_2, t)$ vanishes and the difference between the responses is a maximum, indicating maximum streaming, and 1, when the responses are equal and maximally coherent. Values between these limits are interpreted as the *degree of coherence*, analogous to the probability of human subjects making a judgement of coherence (Anstis, 1985), (Beauvois, 1991).

## 3   RESULTS

Experiments exploring the effect of frequency interval and tone presentation rate and streaming are described in (Beauvois, 1991). Subjects were required to listen to an alternating sequence of tones, ABABAB... for 15 seconds, and then to judge whether at the end of the sequence they perceived an oscillating, trill-like, temporally coherent sequence, or two separate streams, one of interrupted high tones, the other of interrupted

low tones. Their results showed clearly an increasing tendency towards stream segmentation both with increasing frequency difference between A and B, and increasing tone presentation rate, results the model manages substantially to reproduce; as may be seen in figure 2.

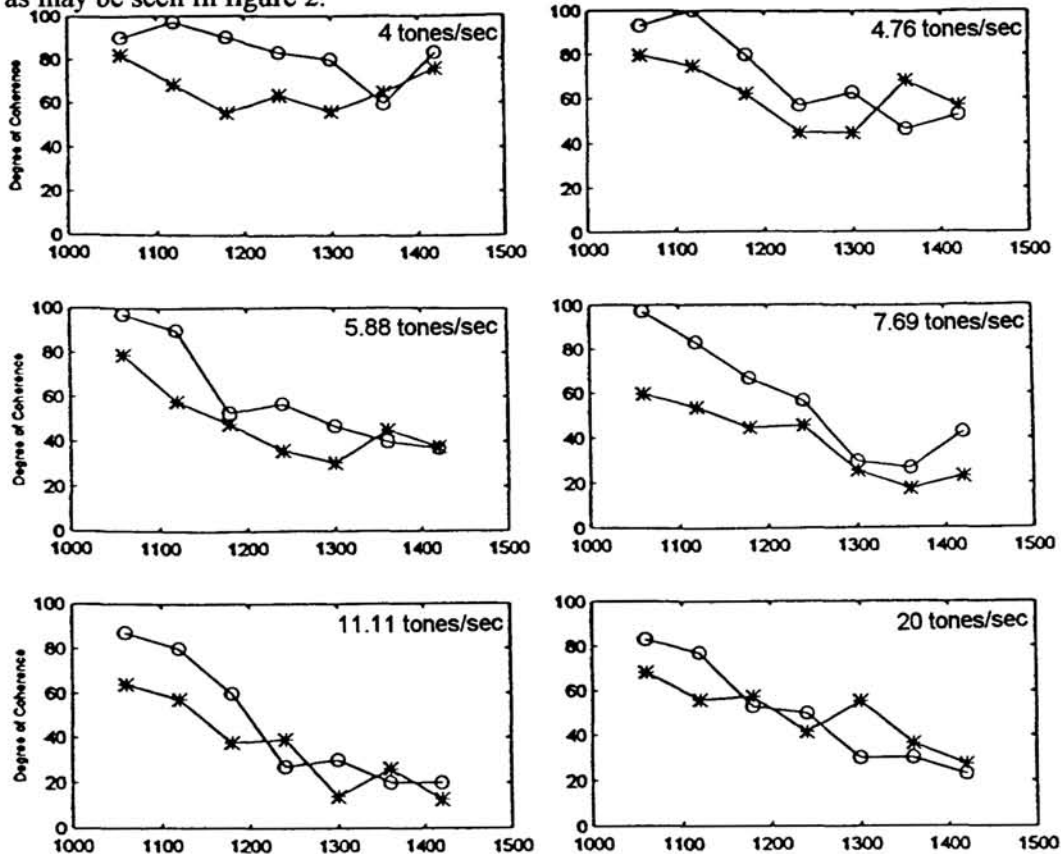

Figure 2 : Mean Psychophysical *'o'* and Model *'*'* Responses to the Stimulus ABAB... (A=1000 Hz, B as indicated along X axis (Hz), tone presentation rates, as shown.)

In investigating the temporal development of stream segmentation, (Anstis, 1985) used a similar stimulus to the experiment described above, but in this case subjects were required to indicate continuously whether they were perceiving a coherent or streaming signal. As can be seen in figure 3, the model clearly reproduces the principal features found in their experiments, i.e. the probability of hearing a single, fused, stream declines during each run, the more rapid the tone presentation rate, the quicker stream segmentation occurs, and the judgements made were quite variable during each run.

In an experiment to investigate whether the organisation of the background sounds affects the foreground, subjects were required to judge whether tone A was higher or lower than B (Bregman, 1975). This judgement was easy when the two tones were presented in isolation, but performance degraded significantly when the distractor tones, X, were included. However, when a series of 'captor' tones, C, with frequency close to X were added, the judgement became easier, and the degree of improvement was inversely related to the difference in frequency between X and C. In the experiment, subjects received an initial priming AB stimulus, followed by a set of 9 tones : CCCXABXCC. The frequency of the captor tones, was manipulated to investigate how the proximity of 'captor' to 'distractor' tones affected the required AB order judgement.

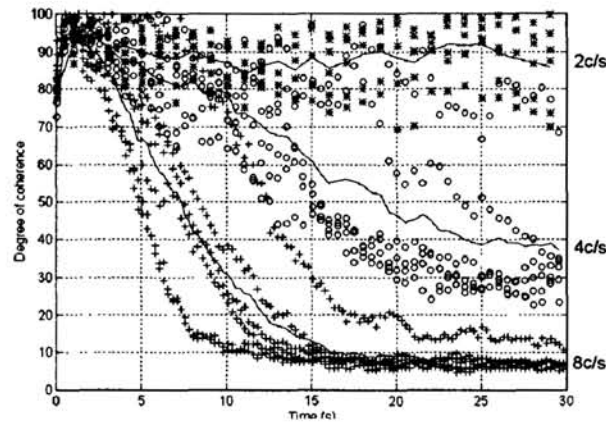

Figure 3 : The Probability of Perceptual Coherence as a Function of Time in Response to Two Alternating Tones. Symbols: '*' 2 tones/s, 'o' 4 tones/s, '+' 8 tones/s.

In order to model this experiment and the effect of priming, an 'attentive' input, focussed on the region of the map corresponding to the A and B tones, was included. We assume, as argued by Bregman, that subjects' performance in this task is related to the degree to which they are able to stream the AB pair separately. His $D$ parameter is a measure of the degree to which AB/BA can be discriminated. The model's performance is then given by the strength of the foreground response to the AB pair as compared to the distractor tones, and $Coh([A\ B],X)$ is used to measure this difference. The model exhibits a similar sensitivity to the distractor/captor frequency difference to that of human subjects, and it appears that the formation of a coherent background stream allows the model to distinguish the foreground group more clearly.

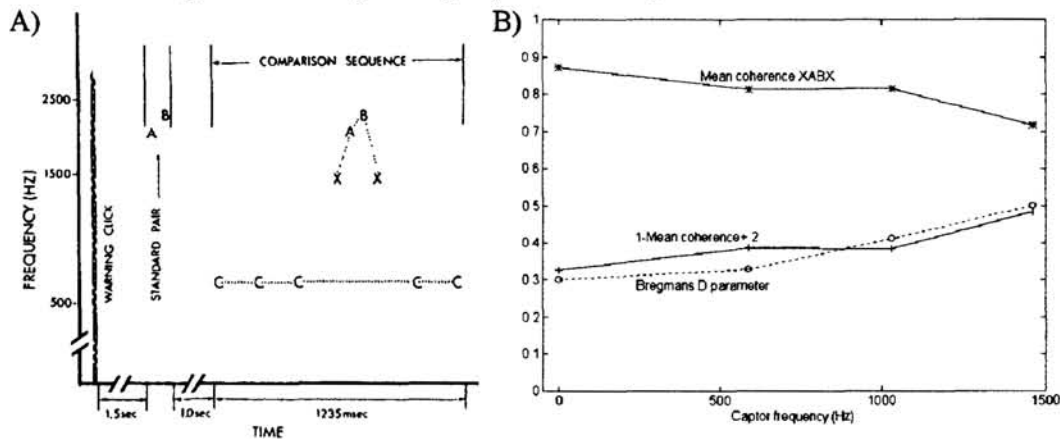

Figure 4 : A) Experiment to Demonstrate the Formation of Multiple Streams, (Bregman, 1975). B) Model Response; '*' Mean Degree of Doherence to XABX, 'o', Bregman's $D$ Parameter, '+' Model's Judgement of Coherence.

## 4  DISCUSSION

The model of streaming which we have presented here is essentially a very simple one, which can, nevertheless, successfully replicate a wide range of psychophysical experiments. Embodied in the model is the idea that the characteristics of the incoming sensory signals result in activity which modifies the way in which subsequent incoming

signals are processed. The inhibitory feedback signals effectively comprise expectations against which later signals are processed. Processing in much of the auditory system seems to be restricted to processing within frequency 'channels'. In this model, it is shown how local interactions, restricted almost entirely to within-channel activity, can form a global computation of stream formation. It is not known where streaming occurs in the auditory system, but feedback projections both within and between nuclei are extensive, perhaps allowing an iterative refinement of streams. Longer range projections, originating from attentive processes or memory, may modify local interactions to facilitate the extraction of recognised or interesting sounds.

The relationship between streaming and frequency interval, could be modelled by systematically graded inhibitory weights between frequency channels. However, in the model this relationship arises directly from the distributed incoming activity patterns, which seems a more robust and plausible solution, particularly if one takes the need to cope with developmental changes into account. Although to simplify the simulations peripheral auditory processing was not included in the model, the activity patterns assumed as input can be produced by the competitive processing of the output from a cochlear model.

An important aspect of intelligent sensory processing is the ability to focus on signals of interest against a background of distracting signals, thereby enabling the perception of significant temporal patterns. Artificial sensory systems, with similar capabilities, could act as robust pre-processors for other systems, such as speech recognisers, fault detection systems, or any other application which required the dynamic extraction and temporal linking of subsets of the overall signal.

## Values Used For Model Parameters

$\alpha=.005$, $c_1=75$, $c_2=100$, $V=[100\ 5\ 5\ 5\ 5]$, $\tau=[.05\ .6\ .6\ .6\ .6]$, $n=2$, $N=100$

## References

Anstis, S., Saida, S., J. (1985) Exptl Psych, 11(3), pp257-271

Beauvois, M.W., Meddis, R. (1991) J. Exptl Psych, 43A(3), pp517-541

Bregman, A.S., Rudnicky, A.I. (1975) J. Exptl Psych , 1(3), pp263-267

Bregman, A.S. (1990) 'Auditory scene analysis', MIT Press

Brown, G.J. (1992) University of Sheffield Research Reports, CS-92-22

Brown, G.J., Cooke, M. (1995) submitted to IJCAI workshop on Computational Auditory Scene Analysis

Cooke, M.P. (1992) Computer Speech and Language 6, pp 153-173

Glasberg, B.R., Moore, B.C.J. (1990) Hearing Research, 47, pp103-138

Llinas, R.R., Pare, D. (1991) Neuroscience, 44(3), pp521-535

Luria, A. (1980) 'Higher cortical functions in man', NY:Basic

van Noorden, L.P.A.S. (1975) doctoral dissertation, published by Institute for Perception Research, PO Box 513, Eindhoven, NL

Wang, D.L. (1995) in 'Handbook of brain theory and neural networks', MIT Press
